# A Second order Cone Programming Formulation for Classifying Missing Data

**Chiranjib Bhattacharyya**
Department of Computer Science and Automation
Indian Institute of Science
Bangalore, 560 012, India
chiru@csa.iisc.ernet.in

**Pannagadatta K. S.**
Department of Electrical Engineering
Indian Institute of Science
Bangalore, 560 012, India
pannaga@ee.iisc.ernet.in

**Alexander J. Smola**
Machine Learning Program
National ICT Australia and ANU
Canberra, ACT 0200, Australia
Alex.Smola@anu.edu.au

## Abstract

We propose a convex optimization based strategy to deal with uncertainty in the observations of a classification problem. We assume that instead of a sample $(\mathbf{x}_i, y_i)$ a distribution over $(\mathbf{x}_i, y_i)$ is specified. In particular, we derive a robust formulation when the distribution is given by a normal distribution. It leads to Second Order Cone Programming formulation. Our method is applied to the problem of missing data, where it outperforms direct imputation.

## 1 Introduction

Denote by $(\mathbf{x}, y) \in \mathcal{X} \times \mathcal{Y}$ patterns with corresponding labels. The typical machine learning formulation only deals with the case where $(\mathbf{x}, y)$ are given *exactly*. Quite often, however, this is not the case — for instance in the case of missing values we may be able (using a secondary estimation procedure) to estimate the values of the missing variables, albeit with a certain degree of uncertainty. It is therefore only natural to take the decreased reliability of such data into account and design estimators accordingly. What we propose in the present paper goes beyond the traditional imputation strategy where missing values are estimated and then used as if they had actually been observed. The key difference in what follows is that we will require that with high probability any $(\tilde{\mathbf{x}}_i, y_i)$ pair, where $\tilde{\mathbf{x}}_i$ is drawn from a distribution of possible $\mathbf{x}_i$, will be estimated correctly. For the sake of simplicity we limit ourselves to the case of binary classification.

The paper is organized as follows: Section 2 introduces the problem of classification with uncertain data. We solve the equations arising in the context of normal random variables in Section 3 which leads to a Second Order Cone Program (SOCP). As an application the problem of classification with missing variables is described in Section 4. We report experimental results in Section 5.

## 2  Linear Classification using Convex Optimization

Assume we have $m$ observations $(\mathbf{x}_i, y_i)$ drawn iid (independently and identically distributed) from a distribution over $\mathcal{X} \times \mathcal{Y}$, where $\mathcal{X}$ is the set of patterns and $\mathcal{Y} = \{\pm 1\}$ are the labels (e.g. the absence/presence of a particular object). It is our goal to find a function $f : \mathcal{X} \to \mathcal{Y}$ which classifies observations $\mathbf{x}$ into classes $+1$ and $-1$.

### 2.1  Classification with Certainty

Assume that $\mathcal{X}$ is a dot product space and $f$ is a linear function

$$f(\mathbf{x}) = \mathrm{sgn}(\langle \mathbf{w}, \mathbf{x} \rangle + b). \tag{1}$$

In the case of linearly separable datasets we can find $(\mathbf{w}, b)$ which separates the two classes. Unfortunately, such separation is not always possible and we need to allow for slack in the separation of the two sets. Consider the formulation

$$\underset{\mathbf{w}, b, \xi}{\text{minimize}} \; \sum_{i=1}^{m} \xi_i \tag{2a}$$

$$\text{subject to } y_i \left( \langle \mathbf{w}, \mathbf{x}_i \rangle + b \right) \geq 1 - \xi_i, \xi_i \geq 0, \|\mathbf{w}\| \leq W \text{ for all } 1 \leq i \leq m \tag{2b}$$

It is well known that this problem minimizes an upper bound on the number of errors. The latter occur whenever $\xi_i \geq 1$, where $\xi_i$ are the slack variables. The Euclidean norm of $\|\mathbf{w}\| = \sqrt{\langle \mathbf{w}, \mathbf{w} \rangle}$, is upper bounded by a user defined constant $W$. This is equivalent to lower bounding the margin, or the separation between the two classes. The resulting discriminant surface is called the *generalized optimal hyperplane* [9]. The statement of (2) is slightly nonstandard. Typically one states the SVM optimization problem as follows [3]:

$$\underset{\mathbf{w}, b, \xi}{\text{minimize}} \; \frac{1}{2} \|\mathbf{w}\|^2 + C \sum_{i=1}^{m} \xi_i \tag{3a}$$

$$\text{subject to } y_i \left( \langle \mathbf{w}, \mathbf{x}_i \rangle + b \right) \geq 1 - \xi_i, \xi_i \geq 0 \text{ for all } 1 \leq i \leq m \tag{3b}$$

Instead of the user defined parameter $W$, the formulation (3) uses another parameter $C$. For a proper choice of $C$ and $W$ the two formulations are equivalent. For the purpose of the present paper, however, (2) will be much more easily amenable to modifications and to cast the resulting problem as a second order cone program (SOCP).

### 2.2  Classification with Uncertainty

So far we assumed that the $(\mathbf{x}_i, y_i)$ pairs are known with certainty. We now relax this to the assumption that we only have a distribution over the $\mathbf{x}_i$, that is $(\mathbf{P}_i, y_i)$ at our disposition (due to a sampling procedure, missing variables, etc.). Formally $\mathbf{x}_i \sim \mathbf{P}_i$. In this case it makes sense to replace the constraints (2b) of the optimization problem (2) by

$$\text{subject to } \Pr\{y_i \left( \langle \mathbf{w}, \mathbf{x}_i \rangle + b \right) \geq 1 - \xi_i\} \geq \kappa_i, \xi_i \geq 0, \|\mathbf{w}\| \leq W \; \forall \; 1 \leq i \leq m \tag{4}$$

Here we replaced the linear classification constraint by a probabilistic one, which is required to hold with probability $\kappa_i \in (0, 1]$. This means that by choosing a value of $\kappa_i$ close to 1 we can find a conservative classifier which will classify even very infrequent $(\mathbf{x}_i, y_i)$ pairs correctly. Hence $\kappa_i$ provides robustness of the estimate with respect to deviating $\mathbf{x}_i$.

It is clear that unless we impose further restrictions on $\mathbf{P}_i$, it will be difficult to minimize the objective $\sum_{i=1}^{m} \xi_i$ with the constraints (4) efficiently. In the following we will consider the special cases of gaussian uncertainty for which a mathematical programming formulation can be found.

## 3 Normal Distributions

For the purpose of this section we assume that $\mathbf{P}_i = \mathcal{N}(\bar{x}_i, \Sigma_i)$, i.e., $\mathbf{x}_i$ is drawn from a Gaussian distribution with mean $\bar{x}_i$ and covariance $\Sigma_i$. We will not require that $\Sigma_i$ has full rank. This means that the uncertainty about $\mathbf{x}_i$ may be limited to individual coordinates or to a subspace of $\mathcal{X}$. As we shall see, this problem can be posed as SOCP.

### 3.1 Robust Classification

Under the above assumptions, the probabilistic constraint (4) becomes

$$\text{subject to } \Pr\{y_i(\langle \mathbf{w}, \mathbf{x}_i \rangle + b) \geq 1 - \xi_i\} \geq \kappa_i \text{ where } \mathbf{x}_i \sim \mathcal{N}(\bar{x}_i, \Sigma_i) \tag{5a}$$

$$\xi_i \geq 0, \|\mathbf{w}\| \leq W \text{ for all } 1 \leq i \leq m \tag{5b}$$

The stochastic constraint can be restated as a deterministic optimization problem

$$\Pr\left\{\frac{z_i - \bar{z}_i}{\sigma_{z_i}} \geq \frac{y_i b + \xi_i - 1 - \bar{z}_i}{\sigma_{z_i}}\right\} \leq \kappa_i \tag{6}$$

where $z_i := -y_i \mathbf{w}^\top \mathbf{x}_i$ is a normal random variable with mean $\bar{z}_i$ and variance $\sigma_{z_i}^2 := \mathbf{w}^\top \Sigma_i \mathbf{w}$. Consequently $(z_i - \bar{z}_i)/\sigma_{z_i}$ is a random variable with zero mean and unit variance and we can compute the lhs of (6) by evaluating the cumulative distribution function for normal distributions

$$\phi(u) := \frac{1}{\sqrt{2\pi}} \int_{-\infty}^{u} e^{-\frac{s^2}{2}} ds.$$

In summary, (6) is equivalent to the condition

$$\phi\left(\frac{y_i b + \xi_i - 1 - \bar{z}_i}{\sigma_{z_i}}\right) \geq \kappa_i.$$

which can be solved (since $\phi(u)$ is monotonic and invertible), for the argument of $\phi$ and obtain a condition on its argument

$$y_i(\mathbf{w}^\top \bar{x}_i + b) \geq 1 - \xi_i + \gamma_i \sqrt{\mathbf{w}^T \Sigma_i \mathbf{w}} \ , \ \gamma_i = \phi^{-1}(\kappa_i) \tag{7}$$

We now proceed to deriving a mathematical programming formulation.

### 3.2 Second Order Cone Programming Formulation

Depending on $\gamma_i$ we can distinguish between three different cases. First consider the case where $\gamma_i = 0$ or $\kappa_i = 0.5$. This means that the second order cone part of the constraint (7) reduces to the linear inequality of (2b). In other words, we recover the linear constraint of a standard SVM.

Secondly consider the case $\gamma_i < 0$ or $\kappa_i < 0.5$. This means that the constraint (7) describes a concave set, which turns the linear classification task into a hard optimization problem. However, it is not very likely that anyone would like to impose such constraints which hold only with low probability. After all, uncertain data requires the constraint to become more restrictive in holding not only for a guaranteed point $x_i$ but rather for an entire set.

Lastly consider the case $\gamma_i > 0$ or $\kappa_i > 0.5$ second order cone constraint. In this case (7) describes a convex set in in $\mathbf{w}, b, \xi_i$. We obtain the following optimization problem:

$$\underset{\mathbf{w}, b, \xi}{\text{minimize}} \sum_{i=1}^{m} \xi_i \tag{8a}$$

$$\text{subject to } y_i(\mathbf{w}^\top \mathbf{x}_i + b) \geq 1 - \xi_i + \gamma_i \|\Sigma_i^{\frac{1}{2}} \mathbf{w}\| \text{ and } \xi_i \geq 0 \ \forall \ 1 \leq i \leq m \tag{8b}$$

$$\|\mathbf{w}\| \leq W \tag{8c}$$

These problems can be solved efficiently by publicly available codes: recent advances in Interior point methods for convex nonlinear optimization [8] have made such problems feasible. As a special case of convex nonlinear optimization SOCPs have gained much attention in recent times. For a further discussion of efficient algorithms and applications of SOCP see [6].

### 3.3 Worst Case Prediction

Note that if at optimality $\xi_i > 0$, the hyperplane intersects with the constraint set $\mathbf{B}(\mathbf{x}_i, \Sigma_i, \gamma_i)$. Moreover, at a later stage we will need to predict the class label to asses on which side of the hyperplane $\mathbf{B}$ lies. If the hyperplane intersects $\mathbf{B}$ we will end up with different predictions for points in the different half spaces. In such a scenario a worst case prediction, $y$ can be

$$y = \mathrm{sgn}(z)\,\mathrm{sgn}(h - \gamma) \text{ where } \gamma = \phi^{-1}(\kappa),\ z = \frac{\langle \mathbf{w}, \mathbf{x}_i \rangle + b}{\sqrt{\mathbf{w}^\top \Sigma \mathbf{w}}} \text{ and } h = |z|. \quad (9)$$

Here $\mathrm{sgn}(z)$ gives us the sign of the point in the center of the ellipsoid and $(h - \gamma)$ is the distance of $z$ from the center. If the hyperplane intersects the ellipsoid, the worst case prediction is then the prediction for all points which are in the opposite half space of the center ($\mathbf{x}_i$). Plugging $\kappa = 0.5$, i.e., $\gamma = 0$ into (9) yields the standard prediction (1). In such a case $h$ can serve as a measure of confidence as to how well the discriminating hyperplane classifies the mean($\mathbf{x}_i$) correctly.

### 3.4 Set Constraints

The same problem as (8) can also be obtained by considering that the uncertainty in each datapoint is characterized by an ellipsoid

$$\mathbf{B}(\mathbf{x}_i, \Sigma_i, \gamma_i) = \{\mathbf{x} : (\mathbf{x} - \mathbf{x}_i)^\top \Sigma_i^{-1} (\mathbf{x} - \mathbf{x}_i) \leq \gamma_i^2\} \quad (10)$$

in conjunction with the constraint

$$y_i \left(\langle \mathbf{w}, \mathbf{x} \rangle + b\right) \geq 1 - \xi_i \text{ for all } \mathbf{x} \in S_i \quad (11)$$

where $S_i = \mathbf{B}(\mathbf{x}_i, \Sigma_i, \gamma_i)$ As before $\gamma_i = \phi^{-1}(\kappa_i)$ for $\kappa_i \geq 0$. In other words, we have $\xi_i = 0$ only when the hyperplane $\mathbf{w}^\top \mathbf{x} + b = 0$ does not intersect the ball $\mathbf{B}(\mathbf{x}_i, \Sigma_i, \gamma_i)$.

Note that this puts our optimization setting into the same category as the knowledge-based SVM, and SDP for invariances as all three deal with the above type of constraint (11). More to the point, in [5] $S_i = S(\mathbf{x}_i, \beta)$ is a polynomial in $\beta$ which describes the set of invariance transforms of $\mathbf{x}_i$ (such as distortion or translation). [4] define $S_i$ to be a polyhedral "knowledge" set, specified by the intersection of linear constraints.

Such considerations suggest yet another optimization setting: instead of specifying a polyhedral set $S_i$ by constraints we can also specify it by its vertices. In particular, we may set $S_i$ to be the convex hull of a set as in $S_i = \mathrm{co}\{\mathbf{x}_{ij} \text{ for } 1 \leq j \leq m_i\}$. By the convexity of the constraint set itself it follows that a necessary and sufficient condition for (11) to hold is that the inequality holds for all $\mathbf{x} \in \{\mathbf{x}_{ij} \text{ for } 1 \leq j \leq m_i\}$. Consequently we can replace (11) by $y_i \left(\langle \mathbf{w}, \mathbf{x}_{ij} \rangle + b\right) \geq 1 - \xi_i$ Note that the index ranges over $j$ rather than $i$. Such a setting allows us to deal with uncertainties, e.g. regarding the range of variables, which are just given by interval boundaries, etc. The table below summarizes the five cases:

| Name | Set $S_i$ | Optimization Problem |
|---|---|---|
| Plain SVM[3] | $\{\mathbf{x}_i\}$ | Quadratic Program |
| Knowledge Based SVM[4] | Polyhedral set | Quadratic Program |
| Invariances [5] | trajectory of polynomial | Semidefinite Program |
| Normal Distribution | $\mathbf{B}(\mathbf{x}_i, \Sigma_i, \gamma_i)$ | Second Order Cone Program |
| Convex Hull | $\mathrm{co}\{\mathbf{x}_{ij} \,\forall\, 1 \leq j \leq m_i\}$ | Quadratic Program |

Clearly all the above constraints can be mixed and matched and it is likely that there will be more additions to this table in the future. More central is the notion of stating the problems via (11) as a starting point.

## 4 Missing Variables

In this section we discuss how to address the missing value problem. Key is how to obtain estimates of the uncertainty in the missing variables. Since our optimization setting allows for uncertainty in terms of a normal distribution we attempt to estimate the latter directly. In other words, we assume that $x|y$ is jointly normal with mean $\mu^y$ and covariance $\Sigma^y$. Hence we have the following two-stage procedure to deal with missing variables:

- Estimate $\Sigma^y, \mu^y$ from incomplete data, e.g. by means of the EM algorithm.
- Use the conditionally normal estimates of $x_{\text{missing}}|(x_{\text{observed}}, y)$ in the optimization problem. This can then be cast in terms of a SOCP as described in the previous section.

Note that there is nothing to prevent us from using other estimates of uncertainty and use e.g. the polyhedral constraints subsequently. However, for the sake of simplicity we focus on normal distributions in this paper.

### 4.1 Estimation of the model parameters

We now detail the computation of the mean and covariance matrices for the datapoints which have missing values. We just sketch the results, for a detailed derivation see e.g. [7].

Let $\mathbf{x} \in R^d$, where $\mathbf{x}_a \in R^{d_a}$ be the vector whose values are known, while $\mathbf{x}_m \in R^{d-d_a}$ be the vector consisting of missing variables. Assuming a jointly normal distribution in $\mathbf{x}$ with mean $\mu$ and covariance $\Sigma$ it follows that

$$\mathbf{x}_m|\mathbf{x}_a \sim \mathcal{N}(\mu_m + \Sigma_{am}\Sigma_{aa}^{-1}(x_a - \mu_a), \Sigma_{mm} - \Sigma_{am}^\top\Sigma_{aa}^{-1}\Sigma_{am}). \tag{12}$$

Here we decomposed $\mu, \Sigma$ according to $(x_a, x_m)$ into

$$\mu = (\mu_a, \mu_m) \text{ and } \Sigma = \left[ \begin{array}{cc} \Sigma_{aa} & \Sigma_{am} \\ \Sigma_{am}^\top & \Sigma_{mm} \end{array} \right]. \tag{13}$$

Hence, knowing $\Sigma, \mu$ we can estimate the missing variables and determine their degree of uncertainty. One can show that [7] to obtain $\Sigma, \mu$ the EM algorithm reads as follows:

1. Initialize $\Sigma, \mu$.
2. Estimate $\mathbf{x}_m|\mathbf{x}_a$ for all observations using (12).
3. Recompute $\Sigma, \mu$ using the completed data set and go to step 2.

### 4.2 Robust formulation for missing values

As stated above, we model the missing variables as Gaussian random variables, with its mean and covariance given by the model described in the previous section. The standard practice for imputation is to discard the covariance and treat the problem as a deterministic problem, using the mean as surrogate. But using the robust formulation (8) one can as well account for the covariance.

Let $m_a$ be number of datapoints for which all the values are available, while $m_m$ be the number of datapoints containing missing values. Then the final optimization problem reads

as follows:

$$\underset{\mathbf{w},b,\xi}{\text{minimize}} \sum_{i=1}^{m} \xi_i \qquad\qquad\qquad\qquad\qquad\qquad (14)$$

$$\text{subject to } y_i\left(\langle \mathbf{w}, \mathbf{x}_i \rangle + b\right) \geq 1 - \xi_i \qquad\qquad \forall 1 \leq i \leq m_a$$

$$y_j(\mathbf{w}^\top \mathbf{x}_j + b) \geq 1 - \xi_j + \phi^{-1}(\kappa_j)\|\Sigma_j^{\frac{1}{2}}\mathbf{w}\| \quad \forall m_a + 1 \leq j \leq m_a + m_m$$

$$\xi_i \geq 0 \qquad\qquad\qquad\qquad\qquad\qquad \forall 1 \leq i \leq m_a + m_m$$

$$\|\mathbf{w}\| \leq W$$

The mean $\mathbf{x}_j$ has two components; $\mathbf{x}_{aj}$ has values available, while the imputed vector is given by $\hat{\mathbf{x}}_{mj}$, via (12). The matrix $\Sigma_j$ has all entries zero except those involving the missing values, given by $C_j$, computed via (12).

The formulation (14) is an optimization problem which involves minimizing a linear objective over linear and second order cone constraints. At optimality the values of $\mathbf{w}, b$, can be used to define a classifier (1). The resulting discriminator can be used to predict the the class label of a test datapoint having missing variables by a process of conditional imputation as follows.

Perform the imputation process assuming that the datapoint comes from class 1(class with label $y = 1$). Specifically compute the mean and covariance, as outlined in section 4.1, and denote them by $\mu_1$ and $\Sigma_1$ (see (13)) respectively. The training dataset of class 1 is to be used in the computation of $\mu_1$ and $\Sigma_1$. Using the estimated $\mu_1$ and $\Sigma_1$ compute $h$ as defined in (9), and denote it by $h_1$. Compute the label of $\mu_1$ with the rule (1), call it $y_1$.

Assuming that the test data comes from class 2 (with label $y = -1$) redo the entire process and denote the resulting mean, covariance, and $h$ by $\mu_2, \Sigma_2, h_2$ respectively. Denote by $y_2$ the label of $\mu_2$ as predicted by (1). We decide that the observation belongs to class with label $y_\mu$ as

$$y_\mu = y_2 \text{ if } h_1 < h_2 \text{ and } y_\mu = y_1 \text{ otherwise} \qquad\qquad (15)$$

The above rule chooses the prediction with higher $h$ value or in other words the classifier chooses the prediction about which it is more confident. Using $y_\mu, h_1, h_2$ as in (15), the worst case prediction rule (9) can be modified as follows

$$y = y_\mu \operatorname{sgn}(h - \gamma) \text{ where } \gamma = \phi^{-1}(\kappa) \text{ and } h = \max(h_1, h_2) \qquad\qquad (16)$$

It is our hypothesis that the formulation (14) along with this decision rule is robust to uncertainty in the data.

## 5 Experiments with the Robust formulation for missing values

Experiments were conducted to evaluate the proposed formulation (14), against the standard imputation strategy. The experiment methodology consisted of creating a dataset of missing values from a completely specified dataset. The robust formulation (14) was used to learn a classifier on the dataset having missing values. The resulting classifier was used to give a worst case prediction (16), on the test data. Average number of disagreements was taken as the error measure. In the following we describe the methodology in more detail.

Consider a fully specified dataset, $\mathcal{D} = \{(\mathbf{x}_i, y_i)|\mathbf{x}_i \in \mathbb{R}^d, y_i \in \{\pm 1\} 1 \leq i \leq N\}$ having $N$ observations, each observation is a $d$ dimensional vector ($\mathbf{x}_i$) and labels $y_i$. A certain fraction($f$) of the observations were randomly chosen. For each of the chosen datapoints $d_m(= 0.5d)$ entries were randomly deleted. This then creates a dataset having $N$ datapoints out of which $N_m(= fN, \ 0 \leq f \leq 1)$ of them have missing values. This data is then

randomly partitioned into test set and training set in the ratio $1 : 9$ respectively. We do this exercise to generate 10 different datasets and all our results are averaged over them.

Assuming that the conditional probability distribution of the missing variables given the other variables is a gaussian, the mean($\mathbf{x}_j$) and the covariance ($\hat{\mathbf{C}}_j$) can be estimated by the methods described in (4.1). The robust optimization problem was then solved for different values of $\kappa$. The parameter $\kappa_j(=\kappa)$ is set to the same value for all the $N_m$ datapoints. For each value of $\kappa$ the worst case error is recorded.

Experimental results are reported for three public domain datasets downloaded from uci repository ([2]). Pima($N = 768, d = 8$), Heart ($N = 270, d = 13$), and Ionosphere($N = 351, d = 34$), were used for experiments.

Setting $\kappa = 0.5$, yields the generalized optimal hyperplane formulation, (2). The generalized optimal hyperplane will be referred to as the nominal classifier. The nominal classifier considers the missing values are well approximated by the mean ($\mathbf{x}_j$), and there is no uncertainty.

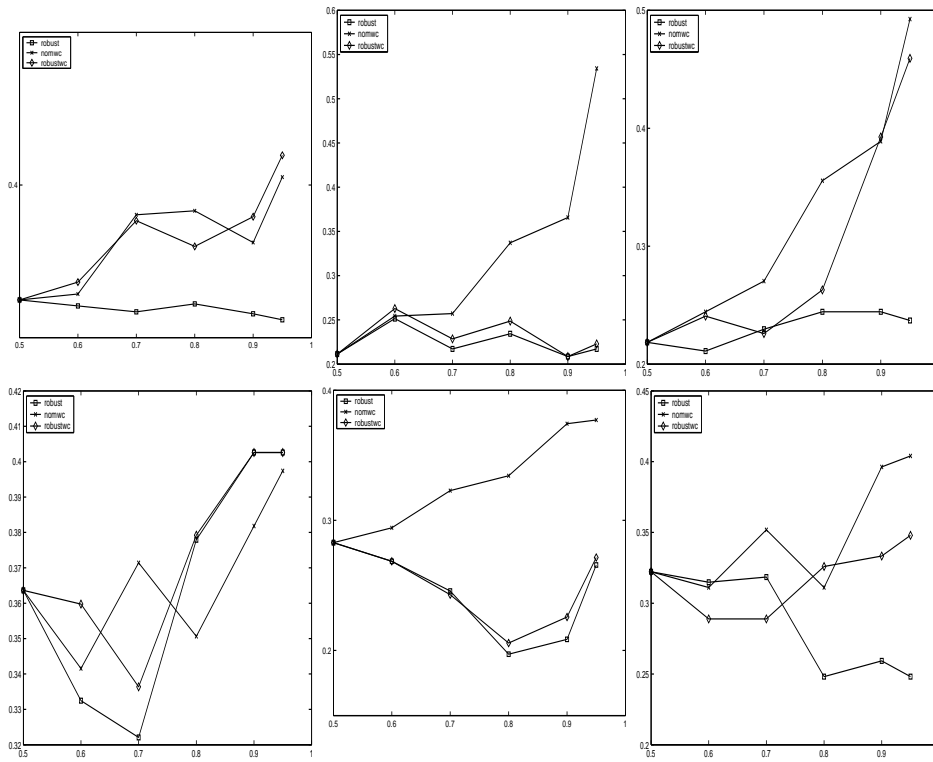

Figure 1: Performance of the robust programming solution for various datasets of the UCI database. From left to right: Pima, Ionosphere, and Heart dataset. Top: small fraction of data with missing variables ($50\%$), Bottom: large number of observations with missing variables ($90\%$)

The experimental results are summarized by the graphs(1). The robust classifier almost always outperforms the nominal classifier in the worst case sense (compare *nomwc* and *robustwc*). Results are presented for low($f = 0.5$), and high ($f = 0.9$) number of missing values. The results show that for low number of missing values($f = 0.5$) the robust classifier is marginally better than the nominal classifier the gain but for large $f = 0.9$ the gain

is significant. This confirms that the imputation strategy fails for high noise.

The standard misclassification error for the robust classifier, using the standard prediction (1), is also shown in the graph with the legend *robust*. As expected the robust classifier performance does not deteriorate in the standard misclassification sense as $\kappa$ is increased.

In summary the results seems to suggest that for low noise level the nominal classifier trained on imputed data performs as good as the robust formulation. But for high noise level the robust formulation yields dividends in the worst case sense.

## 6  Conclusions

An SOCP formulation was proposed for classifying noisy observations and the resulting formulation was applied to the missing data case. In the worst case sense the classifier shows a better performance over the standard imputation strategy. Closely related to this work is the Total Support Vector Classification(TSVC) formulation, presented in [1]. The TSVC formulation tries to reconstruct the original maximal margin classifier in the presence of noisy data. Both TSVC formulation and the approach in this paper address the issue of uncertainty in input data and it would be an important research direction to compare the two approaches.

**Acknowledgements**   CB was partly funded by ISRO-IISc Space technology cell (Grant number IST/ECA/CB/152). National ICT Australia is funded through the Australian Government's *Backing Australia's Ability* initiative, in part through the Australian Research Council. AS was supported by grants of the ARC. We thank Laurent ElGhaoui, Michael Jordan, Gunnar Rätsch, and Frederik Schaffalitzky for helpful discussions and comments.

## References

[1] J. Bi and T. Zhang. Support vector classification with input data uncertainty. In *Advances in Neural Information Processing Systems*. MIT Press, 2004.

[2] C. L. Blake and C. J. Merz. UCI repository of machine learning databases, 1998.

[3] C. Cortes and V. Vapnik. Support vector networks. *Machine Learning*, 20:273–297, 1995.

[4] G. Fung, O. L. Mangasarian, and Jude Shavlik. Knowledge-based support vector machine classifiers. In *Advances in Neural Information Processing Systems*. MIT Press, 2002.

[5] Thore Graepel and Ralf Herbrich. Invariant pattern recognition by semidefinite programming machines. In *Advances in Neural Information Processing Systems 16*, Cambridge, MA, 2003. MIT Press.

[6] M.S. Lobo, L. Vandenberghe, S. Boyd, and H. Lebret. Applications of second-order cone programming. *Linear Algebra and its Applications*, 284(1–3):193–228, 1998.

[7] K. V. Mardia, J. T. Kent, and J. M. Bibby. *Multivariate Analysis*. Academic Press, 1979.

[8] Y. Nesterov and A. Nemirovskii. *Interior Point Algorithms in Convex Programming*. Number 13 in Studies in Applied Mathematics. SIAM, Philadelphia, 1993.

[9] V. Vapnik. *The Nature of Statistical Learning Theory*. Springer, New York, 1995.